# A Hodgkin-Huxley Type Neuron Model That Learns Slow Non-Spike Oscillation

Kenji Doya*          Allen I. Selverston          Peter F. Rowat
Department of Biology
University of California, San Diego
La Jolla, CA 92093-0357, USA

## Abstract

A gradient descent algorithm for parameter estimation which is similar to those used for continuous-time recurrent neural networks was derived for Hodgkin-Huxley type neuron models. Using membrane potential trajectories as targets, the parameters (maximal conductances, thresholds and slopes of activation curves, time constants) were successfully estimated. The algorithm was applied to modeling slow non-spike oscillation of an identified neuron in the lobster stomatogastric ganglion. A model with three ionic currents was trained with experimental data. It revealed a novel role of A-current for slow oscillation below -50 mV.

## 1   INTRODUCTION

Conductance-based neuron models, first formulated by Hodgkin and Huxley [10], are commonly used for describing biophysical mechanisms underlying neuronal behavior. Since the days of Hodgkin and Huxley, tens of new ionic channels have been identified [9]. Accordingly, recent H-H type models have tens of variables and hundreds of parameters [1, 2]. Ideally, parameters of H-H type models are determined by voltage-clamp experiments on individual ionic currents. However, these experiments are often very difficult or impossible to carry out. Consequently, many parameters must be hand-tuned in computer simulations so that the model behavior resembles that of the real neuron. However, a manual search in a high dimensional

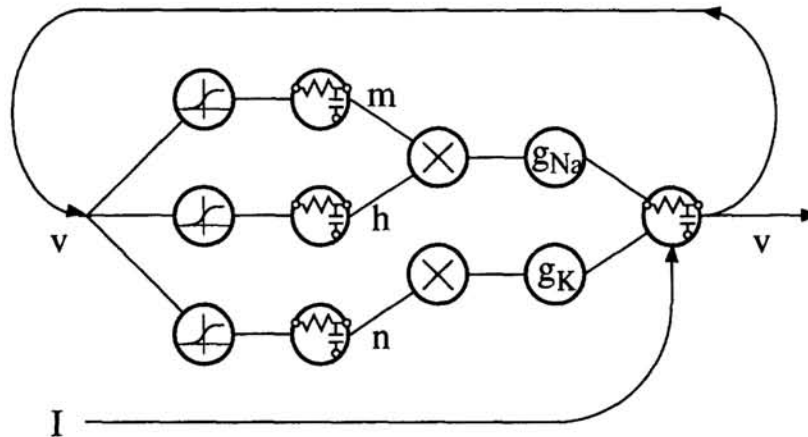

Figure 1: A connectionist's view of the H-H neuron model.

parameter space is very unreliable. Moreover, even if a good match is found between the model and the real neuron, the validity of the parameters is questionable because there are, in general, many possible settings that lead to apparently the same behavior.

We propose an automatic parameter tuning algorithm for H-H type neuron models [5]. Since a H-H type model is a network of sigmoid functions, multipliers, and leaky integrators (Figure 1), we can tune its parameters in a manner similar to the tuning of connection weights in continuous-time neural network models [6, 12]. By training a model from many initial parameter points to match the experimental data, we can systematically estimate a region in the parameter space, instead of a single point.

We first test if the parameters of a spiking neuron model can be identified from the membrane potential trajectories. Then we apply the learning algorithm to a model of slow non-spike oscillation of an identified neuron in the lobster stomatogastric ganglion [7]. The resulting model suggests a new role of A-current [3] for slow oscillation in the membrane potential range below -50 mV.

## 2   STANDARD FORM OF IONIC CURRENTS

Historically, different forms of voltage dependency curves have been used to represent the kinetics of different ionic channels. However, in order to derive a simple, efficient learning algorithm, we chose a unified form of voltage dependency curves which is based on statistical physics of ionic channels [11] for all the ionic currents in the model.

The dynamics of the membrane potential $v$ is given by

$$C\dot{v} = I - \sum_j I_j, \quad I_j = g_j a_j^{p_j} b_j^{q_j}(v - v_{rj}), \tag{1}$$

where $C$ is the membrane capacitance and $I$ is externally injected current. The $j$-th ionic current $I_j$ is the product of the maximum conductance $g_j$, activation variable

$a_j$, inactivation variable $b_j$, and the difference of the membrane potential $v$ from the reversal potential $v_{rj}$. The exponents $p_j$ and $q_j$ represent multiplicity of gating elements in the ionic channels and are usually an integer between 0 and 4. Variables $a_j$ and $b_j$ are assumed to obey the first order differential equation

$$\dot{x} = k_x(v) \cdot (-x + x_\infty(v)), \quad (x = a_j, b_j). \tag{2}$$

Their steady states $a_{j\infty}$ and $b_{j\infty}$ are sigmoid functions of the membrane potential

$$x_\infty(v) = \frac{1}{1 + e^{-s_x(v - v_x)}}, \quad (x = a_j, b_j), \tag{3}$$

where $v_x$ and $s_x$ represent the threshold and slope of the steady state curve, respectively. The rate coefficients $k_{a_j}(v)$ and $k_{b_j}(v)$ have the voltage dependence [11]

$$k_x(v) = \frac{1}{t_x} \cosh \frac{s_x(v - v_x)}{2}, \quad (x = a_j, b_j), \tag{4}$$

where $t_x$ is the time constant.

## 3   ERROR GRADIENT CALCULUS

Our goal is to minimize the average error over a cycle with period $T$:

$$E = \frac{1}{T} \int_0^T \frac{1}{2}(v(t) - v^*(t))^2 dt, \tag{5}$$

where $v^*(t)$ is the target membrane potential trajectory.

We first derive the gradient of $E$ with respect to the model parameters $(..., \theta_i, ...) = (..., g_j, v_{a_j}, s_{a_j}, t_{a_j}, ...)$. In studies of recurrent neural networks, it has been shown that *teacher forcing* is very important in training autonomous oscillation patterns [4, 6, 12, 13]. In H-H type models, teacher forcing drives the activation and inactivation variables by the target membrane potential $v^*(t)$ instead of $v(t)$ as follows.

$$\dot{x} = k_x(v^*(t)) \cdot (-x + x_\infty(v^*(t))) \quad (x = a_j, b_j). \tag{6}$$

We use (6) in place of (2) during training.

The effect of a small change in a parameter $\theta_i$ of a dynamical system

$$\dot{X} = F(X; ..., \theta_i, ...), \quad (X \in \mathbf{R}^n) \tag{7}$$

is evaluated by the variation equation

$$\dot{Y} = \frac{\partial F}{\partial X} Y + \frac{\partial F}{\partial \theta_i}, \quad (Y \in \mathbf{R}^n), \tag{8}$$

which is an $n$-dimensional linear system with time-varying coefficients [6, 12]. In general, this variation calculus requires $O(n^2)$ arithmetics for each parameter. However, in the case of H-H model with teacher forcing, (8) reduces to a first or second order linear system. For example, the effect of a small change in the maximum conductance $g_j$ on the membrane potential $v$ is estimated by

$$C\dot{y} = -G(t)y - a_j(t)^{p_j} b_j(t)^{q_j}(v(t) - v_{rj}), \tag{9}$$

where $G(t) = \sum_k g_k a_k(t)^{p_k} b_k(t)^{q_k}$ is the total membrane conductance. Similarly, the effect of the activation threshold $v_{a_j}$ is estimated by the equations

$$C\dot{y} = -G(t)y - g_j p_j a_j(t)^{p_j-1} b_j(t)^{q_j} (v(t) - v_{rj}) z,$$

$$\dot{z} = -k_{a_j}(t) \left[ z + \frac{s_{a_j}}{2} \{a_j(t) + a_{j\infty}(t) - 2a_j(t)a_{j\infty}(t)\} \right]. \qquad (10)$$

The solution $y(t)$ represents the perturbation in $v$ at time $t$, namely $\frac{\partial v(t)}{\partial \theta_i}$. The error gradient is then given by

$$\frac{\partial E}{\partial \theta_i} = \frac{1}{T} \int_0^T (v(t) - v^*(t)) \frac{\partial v(t)}{\partial \theta_i} dt. \qquad (11)$$

## 4   PARAMETER UPDATE

Basically, we can use arbitrary gradient-based optimization algorithms, for example, simple gradient descent or conjugate gradient descent. The particular algorithm we used was a continuous-time version of gradient descent on normalized parameters.

Because the parameters of a H-H type model have different physical dimensions and magnitudes, it is not appropriate to perform simple gradient descent on them. We represent each parameter by the default value $\bar{\theta}_i$ and the deviation $\tilde{\theta}_i$ as below.

$$g_j = \bar{g}_j e^{\tilde{g}_j}, \quad s_x = \bar{s}_x e^{\tilde{s}_x}, \quad t_x = \bar{t}_x e^{\tilde{t}_x}, \quad v_x = \bar{v}_x + \Delta v \cdot \tilde{v}_x, \quad (x = a_j, b_j). \qquad (12)$$

Then we perform gradient descent on the normalized parameters $\tilde{\theta}_i$.

Instead of updating the parameters in *batches*, i.e. after running the model for $T$ and integrating the error gradient by (11), we updated the parameters *on-line* using the running average of the gradient as follows.

$$T_a \dot{\Delta}_{\tilde{\theta}_i} = -\Delta_{\tilde{\theta}_i} + \frac{1}{T}(v(t) - v^*(t)) \frac{\partial v(t)}{\partial \theta_i} \frac{\partial \theta_i}{\partial \tilde{\theta}_i},$$

$$\dot{\tilde{\theta}}_i = -\varepsilon \Delta_{\tilde{\theta}_i}, \qquad (13)$$

where $T_a$ is the averaging time and $\varepsilon$ is the learning rate. This on-line scheme was less susceptible to $2T$-periodic parameter oscillation than batch update scheme and therefore we could use larger learning rates.

## 5   PARAMETER ESTIMATION OF A SPIKING MODEL

First, we tested if a model with random initial parameters can estimate the parameters of another model by training with its membrane potential trajectories. The default parameters $\bar{\theta}_i$ of the model was set to match the original H-H model [10] (Table 1). Its membrane potential trajectories at five different levels of current injection ($I = 0, 15, 30, 45,$ and $60\mu A/cm^2$) were used alternately as the target $v^*(t)$.

We ran 100 trials after initializing $\tilde{\theta}_i$ randomly in [-0.5,+0.5]. In 83 cases, the error became less than 1.3 mV rms after 100 cycles of training. Figure 2a is an example of the oscillation patterns of the trained model. The mean of the normalized

Table 1: Parameters of the spiking neuron model. Subscripts $L$, $Na$ and $K$ specifies leak, sodium and potassium currents, respectively. Constants: $C{=}1\mu F/cm^2$, $v_{Na}{=}55mV$, $v_K{=}{-}72mV$, $v_L{=}{-}50mV$, $p_{Na}{=}3$, $q_{Na}{=}1$, $p_K{=}4$, $q_K{=}p_L{=}q_L{=}0$, $\Delta v{=}20mV$, $\epsilon{=}0.1$, $T_a = 5T$.

| | default value $\bar{\theta}_i$ | | $\tilde{\theta}_i$ after learning mean | s.d. |
|---|---|---|---|---|
| $g_L$ | 0.3 | mS/cm$^2$ | -0.017 | 0.252 |
| $g_{Na}$ | 120.0 | mS/cm$^2$ | -0.002 | 0.248 |
| $v_{aNa}$ | -36.0 | mV | 0.006 | 0.033 |
| $s_{aNa}$ | 0.1 | 1/mV | -0.052 | 0.073 |
| $t_{aNa}$ | 0.5 | msec | -0.103 | 0.154 |
| $v_{bNa}$ | -62.0 | mV | 0.012 | 0.202 |
| $s_{bNa}$ | -0.09 | 1/mV | -0.010 | 0.140 |
| $t_{bNa}$ | 12.0 | msec | 0.093 | 0.330 |
| $g_K$ | 40.0 | mS/cm$^2$ | 0.050 | 0.264 |
| $v_{aK}$ | -50.0 | mV | -0.021 | 0.136 |
| $s_{aK}$ | 0.06 | 1/mV | -0.061 | 0.114 |
| $t_{aK}$ | 5.0 | msec | -0.073 | 0.168 |

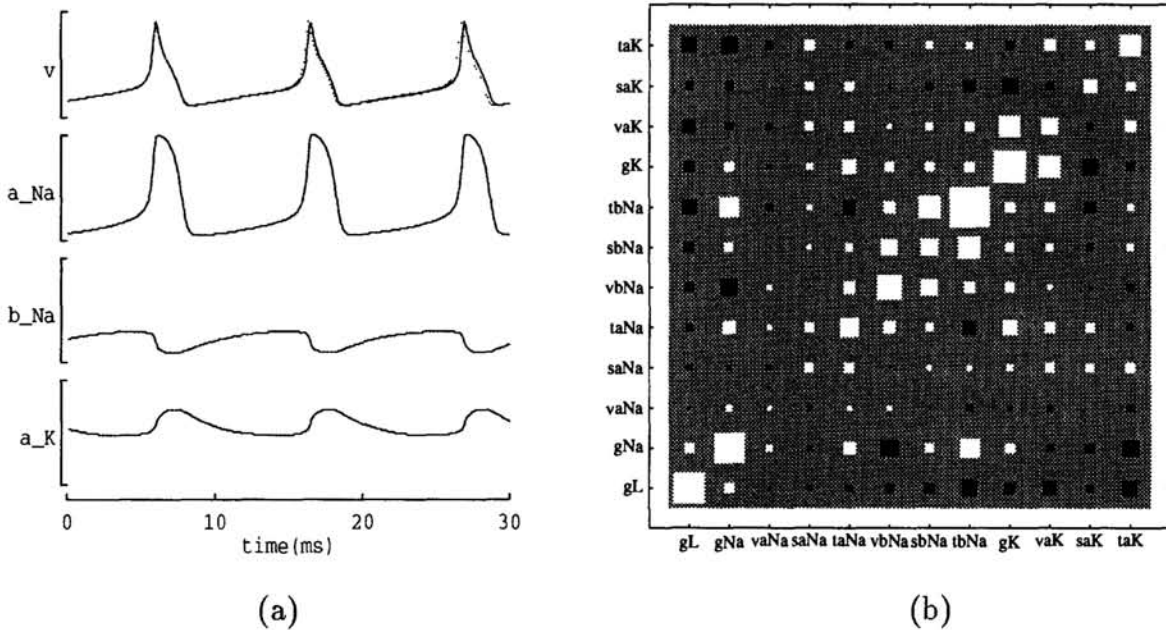

Figure 2: (a) The trajectory of the spiking neuron model at $I = 30\mu A/cm^2$. $v$: membrane potential (-80 to +40 mV). $a$ and $b$: activation and inactivation variables (0 to 1). The dotted line in $v$ shows the target trajectory $v^*(t)$. (b) Covariance matrix of the normalized parameters $\tilde{\theta}_i$ after learning. The black and white squares represent negative and positive covariances, respectively.

Table 2: Parameters of the DG cell model. Constants: $C=1\mu\text{F}/\text{cm}^2$, $v_A=$-80mV, $v_H=$ -10mV, $v_L=$-50mV, $p_A=3$, $q_A=1$, $p_H=1$, $q_H=p_L=q_L=0$, $\Delta v=20$mV, $\epsilon=0.1$, $T_a = 2T$.

|  | $\bar{\theta}_i$ | tuned $\theta_i$ |  |
|---|---|---|---|
| $g_L$ | 0.01 | 0.025 | mS/cm$^2$ |
| $g_A$ | 50 | 41.0 | mS/cm$^2$ |
| $v_{aA}$ | -12 | -11.1 | mV |
| $s_{aA}$ | 0.04 | 0.022 | 1/mV |
| $t_{aA}$ | 7.0 | 7.0 | msec |
| $v_{bA}$ | -62 | -76 | mV |
| $s_{bA}$ | -0.16 | -0.19 | 1/mV |
| $t_{bA}$ | 300 | 292 | msec |
| $g_H$ | 0.1 | 0.039 | mS/cm$^2$ |
| $v_{aH}$ | -70 | -75.1 | mV |
| $s_{aH}$ | -0.14 | -0.11 | 1/mV |
| $t_{aH}$ | 3000 | 4400 | msec |

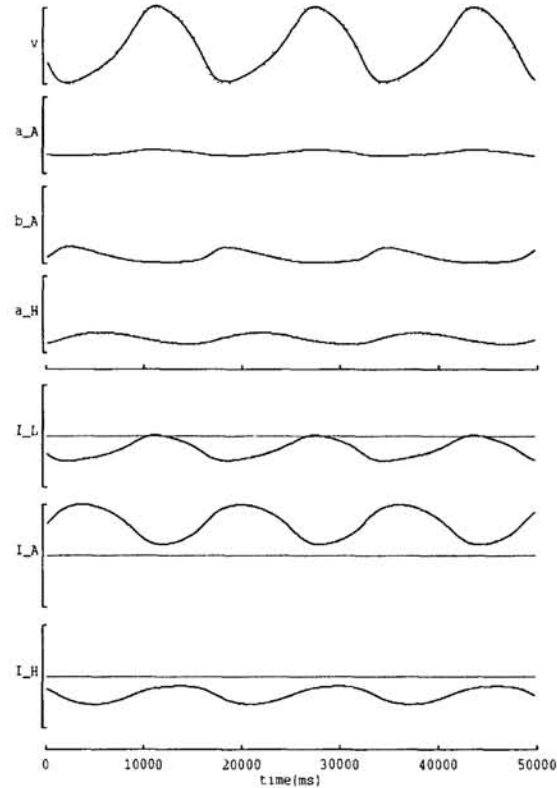

Figure 3: Oscillation pattern of the DG cell model. $v$: membrane potential (-70 to -50 mV). $a$ and $b$: activation and inactivation variables (0 to 1). $I$: ionic currents (-1 to +1 $\mu A/cm^2$).

parameters $\tilde{\theta}_i$ were nearly zero (Table 1), which implies that the original parameter values were successfully estimated by learning. The standard deviation of each parameter indicates how critical its setting is to replicate the given oscillation patterns. From the covariance matrix of the parameters (Figure 2b), we can estimate the distribution of the solution points in the parameter space.

# 6    MODELING SLOW NON-SPIKE OSCILLATION

Next we applied the algorithm to experimental data from the "DG cell" of the lobster stomatogastric ganglion [7]. An isolated DG cell oscillates endogenously with the acetylcholine agonist pilocarpine and the sodium channel blocker TTX. The oscillation period is 5 to 20 seconds and the membrane potential is approximately between -70 and -50 mV. From voltage-clamp data from other stomatogastric neurons [8], we assumed that A-current (potassium current with inactivation) [3] and H-current (hyperpolarization-activated slow inward current) are the principal active currents in this voltage range. The default parameters for these currents were taken from [2] (Table 2).

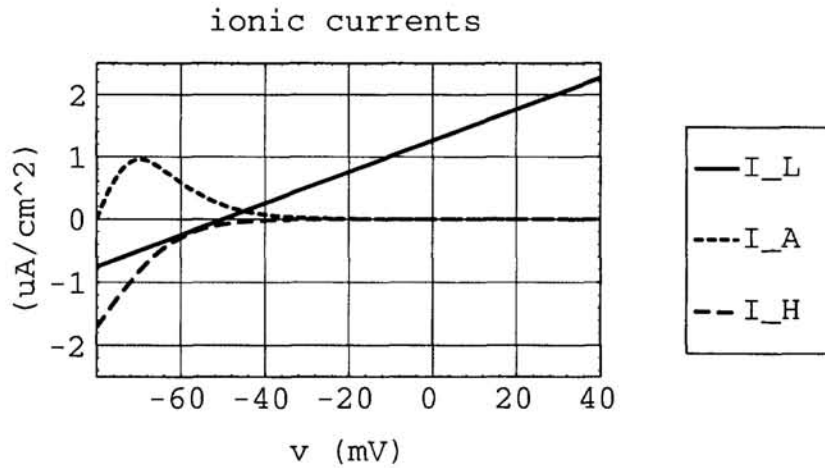

Figure 4: Current-voltage curves of the DG cell model. Outward current is positive.

Figure 3 is an example of the model behavior after learning for 700 cycles. The actual output $v$ of the model, which is shown in the solid curve, was very close to the target output $v^*(t)$, which is shown in the dotted curve. The bottom three traces show the ionic currents underlying this slow oscillation. Figure 4 shows the steady state I-V curves of three currents. A-current has *negative conductance* in the range from -70 to -40 mV. The resulting positive feedback on the membrane potential destabilizes a quiescent state. If we rotate the I-V diagram 180 degrees, it looks similar to the I-V diagram for the H-H model; the faster outward A-current in our model takes the role of the fast inward sodium current in the H-H model and the slower inward H-current takes the role of the outward potassium current.

## 7   DISCUSSION

The results indicate that the gradient descent algorithm is effective for estimating the parameters of H-H type neuron models from membrane potential trajectories.

Recently, an automatic parameter search algorithm was proposed by Bhalla and Bower [1]. They chose only the maximal conductances as free parameters and used conjugate gradient descent. The error gradient was estimated by slightly changing each of the parameters. In our approach, the error gradient was more efficiently derived by utilizing the variation equations. The use of teacher forcing and parameter normalization was essential for the gradient descent to work.

In order for a neuron to be an endogenous oscillator, it is required that a fast positive feedback mechanism is balanced with a slower negative feedback mechanism. The most popular example is the positive feedback by the sodium current and the negative feedback by the potassium current in the H-H model. Another common example is the inward calcium current counteracted by the calcium dependent outward potassium current. We found another possible combination of positive and negative feedback with the help of the algorithm: the *inactivation* of the outward A-current and the activation of the slow inward H-current.

**Acknowledgements**

The authors thank Rob Elson and Thom Cleland for providing physiological data from stomatogastric cells. This study was supported in part by ONR grant N00014-91-J-1720.

**References**

[1] U. S. Bhalla and J. M. Bower. Exploring parameter space in detailed single neuron models: Simulations of the mitral and granule cells of the olfactory bulb. *Journal of Neurophysiology*, 69:1948–1965, 1993.

[2] F. Buchholtz, J. Golowasch, I. R. Epstein, and E. Marder. Mathematical model of an identified stomatogastric ganglion neuron. *Journal of Neurophysiology*, 67:332–340, 1992.

[3] J. A. Connor, D. Walter, and R. McKown. Neural repetitive firing, modifications of the Hodgkin-Huxley axon suggested by experimental results from crustacean axons. *Biophysical Journal*, 18:81–102, 1977.

[4] K. Doya. Bifurcations in the learning of recurrent neural networks. In *Proceedings of 1992 IEEE International Symposium on Circuits and Systems*, pages 6:2777–2780, San Diego, 1992.

[5] K. Doya and A. I. Selverston. A learning algorithm for Hodgkin-Huxley type neuron models. In *Proceedings of IJCNN'93*, pages 1108–1111, Nagoya, Japan, 1993.

[6] K. Doya and S. Yoshizawa. Adaptive neural oscillator using continuous-time back-propagation learning. *Neural Networks*, 2:375–386, 1989.

[7] R. C. Elson and A. I. Selverston. Mechanisms of gastric rhythm generation in the isolated stomatogastric ganglion of spiny lobsters: Bursting pacemaker potential, synaptic interactions, and muscarinic modulation. *Journal of Neurophysiology*, 68:890–907, 1992.

[8] J. Golowasch and E. Marder. Ionic currents of the lateral pyloric neuron of stomatogastric ganglion of the crab. *Journal of Neurophysiology*, 67:318–331, 1992.

[9] B. Hille. *Ionic Channels of Excitable Membranes*. Sinauer, 1992.

[10] A. L. Hodgkin and A. F. Huxley. A quantitative description of membrane currents and its application to conduction and excitation in nerve. *Journal of Physiology*, 117:500–544, 1952.

[11] H. Lecar, G. Ehrenstein, and R. Latorre. Mechanism for channel gating in excitable bilayers. *Annals of the New York Academy of Sciences*, 264:304–313, 1975.

[12] P. F. Rowat and A. I. Selverston. Learning algorithms for oscillatory networks with gap junctions and membrane currents. *Network*, 2:17–41, 1991.

[13] R. J. Williams and D. Zipser. Gradient based learning algorithms for recurrent connectionist networks. Technical Report NU-CCS-90-9, College of Computer Science, Northeastern University, 1990.

## Footnotes

*current address: The Salk Institute, CNL, P.O. Box 85800, San Diego, CA 92186-5800.
